# Competition adds complexity

**Judy Goldsmith**
Department of Computer Science
University of Kentucky
Lexington, KY
goldsmit@cs.uky.edu

**Martin Mundhenk**
Friedrich-Schiller-Universität Jena
Jena, Germany
mundhenk@cs.uni-jena.de

## Abstract

It is known that deterimining whether a DEC-POMDP, namely, a cooperative partially observable stochastic game (POSG), has a cooperative strategy with positive expected reward is complete for NEXP. It was not known until now how cooperation affected that complexity. We show that, for competitive POSGs, the complexity of determining whether one team has a positive-expected-reward strategy is complete for $\text{NEXP}^{\text{NP}}$.

## 1  Introduction

From online auctions to Texas Hold'em, AI is captivated by multi-agent interactions based on competition. The problem of finding a *winning strategy* harks back to the first days of chess programs. Now, we are starting to have the capacity to handle issues like stochastic games, partial information, and real-time video inputs for human player modeling. This paper looks at the complexity of computations involving the first two factors: partially observable stochastic games (POSGs).

There are many factors that could affect the complexity of different POSG models: Do the players, collectively, have sufficient information to reconstruct a state? Do they communicate or cooperate? Is the game zero sum, or do the players' individual utilities depend on other players' utilities? Do the players even have models for other players' utilities?

The ultimate question is, what is the complexity of finding a winning strategy for a particular player, with no assumptions about joint observations or knowledge of other players' utilities. Since a special case of this is the DEC-POMDP, where finding an optimal (joint, cooperative) policy is known to be NEXP-hard [1], this problem cannot be any easier than in NEXP.

We show that one variant of this problem is hard for the class $\text{NEXP}^{\text{NP}}$.

## 2  Definitions and Preliminaries

### 2.1  Partially observable stochastic games

A *partially observable stochastic game* (POSG) describes multi-player stochastic game with imperfect information by its states and the consequences of the players actions on the system. We follow the definition from [2] and denote it as a tuple $\mathcal{M} = (I, S, s_0, A, O, t, o, r)$, where

- $I$ is the finite set $\{1, 2, \ldots, k\}$ of agents (or players), $S$ is a finite set of *states*, with distinguished initial state $s_0 \in S$, $A$ is a finite set of *actions*, and $O$ is a finite set of *observations*
- $t : S \times A^k \times S \rightarrow [0,1]$ is the *transition probability function*, where $t(s, a_1, \ldots, a_k, s')$ is the probability that state $s'$ is reached from state $s$ when each agent $i$ chooses action $a_i$
- $o : S \times I \rightarrow O$ is the *observation function* , where $o(s, i)$ is the observation made in state $s$ by agent $i$, and

- $r : S \times A^k \times I \to \mathbf{Z}$ is the *reward function*, where $r(s,a_1,\ldots,a_k,i)$ is the reward gained by agent $i$ in state $s$, when the agents take actions $a_1,\ldots,a_k$. ($\mathbf{Z}$ is the set of integers.)

A POSG where all agents have the same reward function is called a *decentralized partially-observable Markov decision process* (see [1]).

Let $\mathcal{M} = (I,S,s_0,A,O,t,o,r)$ be a POSG. A *step* of $\mathcal{M}$ is a transition from one state to another according to the transition probability function $t$. A *run* of $\mathcal{M}$ is a sequence of steps that starts in the initial state $s_0$. The outcome of each step is probabilistic and depends on the actions chosen. For each agent, a *policy* describes how to choose actions depending on observations made during the run of the process. A *(history-dependent) policy* $\pi$ chooses an action dependent on all observations made by the agent during the run of the process. This is described as a function $\pi : O^* \to A$, mapping each finite sequence of observations to an action.

A *trajectory* $\theta$ *of length* $|\theta| = m$ *for* $\mathcal{M}$ is a sequence of states $\theta = \sigma_1, \sigma_2, \ldots, \sigma_m$ ($m \geq 1$, $\sigma_i \in S$) which starts with the initial state of $\mathcal{M}$, i.e. $\sigma_1 = s_0$. Given policies $\pi_1,\ldots,\pi_k$, each trajectory $\theta$ has a probability $\mathrm{prob}(\theta, \pi_1, \ldots, \pi_k)$. We will use some abbreviations in the sequel. For $\pi_1, \ldots, \pi_k$ we will write $\pi_1^k$, and for $\pi_1(o(\sigma_1,1) \cdots o(\sigma_j,1)), \ldots, \pi_k(o(\sigma_1,k) \cdots o(\sigma_j,k))$ we will write $\pi_1^k(\theta_1^j)$ accordingly. Then $\mathrm{prob}(\theta, \pi_1, \ldots, \pi_k)$ is defined by

$$\mathrm{prob}(\theta, \pi_1^k) = \prod_{i=1}^{|\theta|-1} t(\sigma_i, \pi_1^k(\theta_1^i), \sigma_{i+1}) \ .$$

We use $T_l(s)$ to denote all length $l$ trajectories which start in the initial state $s_0$ and end in state $s$. The expected reward $R_i(s,l,\pi_1^k)$ obtained by agent $i$ in state $s$ after exactly $l$ steps under policies $\pi_1^k$ is the reward obtained in $s$ by the actions according to $\pi_1^k$ weighted by the probability that $s$ is reached after $l$ steps,

$$R_i(s,l,\pi_1^k) = \sum_{\theta \in T_l(s), \theta = (\sigma_1,\ldots,\sigma_l)} r(s, \pi_1^k(\theta_1^l), i) \cdot \mathrm{prob}(\theta, \pi_1^k) \ .$$

A POSG may behave differently under different policies. The quality of a policy is determined by its *performance*, i.e. by the sum of expected rewards received on it. We use $|\mathcal{M}|$ to denote the size of the representation of $\mathcal{M}$.[1] The *short-term performance for policies* $\pi_1^k$ for agent $i$ with POSG $\mathcal{M}$ is the expected sum of rewards received by agent $i$ during the next $|\mathcal{M}|$ steps by following the policy $\pi_1^k$, i.e.

$$perf_i(\mathcal{M}, \pi_1^k) = \sum_{s \in S} R_i(s, |\mathcal{M}|, \pi_1^k) \ .$$

The performance is also called the expected reward.

Agents may cooperate or compete in a stochastic game. We want to know whether a stochastic game can be won by some agents. This is formally expressed in the following decision problems.

**The cooperative agents problem for *k* agents:**

    instance:    a POSG $\mathcal{M}$ for $k$ agents
    query:    are there policies $\pi_1,\ldots,\pi_k$ under which every agent has positive performance ? (I.e. $\exists \pi_1,\ldots,\pi_k : \bigwedge_{i=1}^{k} perf_i(\mathcal{M}, \pi_1^k) > 0$ ?)

**The competing agents problem for *2k* agents:**

    instance:    a POSG $\mathcal{M}$ for $2k$ agents
    query:    are there policies $\pi_1,\ldots,\pi_k$ under which all agents $1,2,\ldots,k$ have positive performance independent of which policies agents $k+1,k+2,\ldots,2k$ choose? (I.e. $\exists \pi_1,\ldots,\pi_k \forall \pi_{k+1},\ldots,\pi_{2k} : \bigwedge_{i=1}^{k} perf_i(\mathcal{M}, \pi_1^{2k}) > 0$ ?)

It was shown by Bernstein et al. [1] that the cooperative agents problem for two or more agents is complete for NEXP.

## 2.2  NEXP$^{\text{NP}}$

A Turing machine $M$ has exponential running time, if there is a polynomial $p$ such that for every input $x$, the machine $M$ on input $x$ halts after at most $2^{p(|x|)}$ steps. NEXP is the class of sets that can be decided by a nondeterministic Turing machine within exponential time. NEXP$^{\text{NP}}$ is the class of sets that can be decided by a nondeterministic oracle Turing machine within exponential time, when a set in NP is used as an oracle. Similar as for the class NP$^{\text{NP}}$, it turns out that a NEXP$^{\text{NP}}$ computation can be performed by an NEXP oracle machine that asks exactly one query to a co NP oracle and accepts if and only if the oracle accepts.

## 2.3  Domino tilings

Domino tiling problems are useful for reductions between different kinds of computations. They have been proposed by Wang [3], and we will use it according to the following definition.

**Definition 2.1** *We use $[m]$ to denote the set $\{0,1,2,\dots,m-1\}$. A tile type $T=(V,H)$ consists of two finite sets $V,H \subseteq \mathbf{N} \times \mathbf{N}$. A $T$-tiling of an m-square ($m \in \mathbf{N}$) is a mapping $\tau : [m] \times [m] \to \mathbf{N}$ that satisfies both the following conditions.*

1. *Every pair of two neighboured tiles in the same row is in $H$.*

   *I.e. for all $r \in [m]$ and $c \in [m-1]$, $(\tau(r,c),\tau(r,c+1)) \in H$.*

2. *Every pair of two neighboured tiles in the same column is in $V$.*

   *I.e. for all $r \in [m-1]$ and $c \in [m]$, $(\tau(r,c),\tau(r+1,c)) \in V$.*

*The* exponential square tiling problem *is the set of all pairs $(T,1^k)$, where $T$ is a tile type and $1^k$ is a string consisting of $k$ 1s ($k \in \mathbf{N}$), such that there exists a $T$-tiling of the $2^k$-square.*

It was shown by Savelsbergh and van Emde Boas [4] that the exponential square tiling problem is complete for NEXP. We will consider the following variant, which we call the *exponential $\Sigma_2$ square tiling problem*: given a pair $(T,1^k)$, does there exist a row $w$ of tiles and a $T$-tiling of the $2^k$-square with final row $w$, such that there exists no $T$-tiling of the $2^k$-square with initial row $w$? The proof technique of Theorem 2.29 in [4], which translates Turing machine computations into tilings, is very robust in the sense that simple variants of the square tiling problem can analogously be shown to be complete for different complexity classes. Together with the above characterization of NEXP$^{\text{NP}}$ it can be used to prove the following.

**Theorem 2.2** *The exponential $\Sigma_2$ square tiling problem is complete for* NEXP$^{\text{NP}}$.

# 3   Results

POSGs can be seen as a generalization of partially-observable Markov decision processes (PO-MDPs) in that POMDPs have only one agent and POSGs allow for many agents. Papadimitriou and Tsitsiklis [5] proved that it is PSPACE-complete to decide the cooperative agents problem for POMDPs. The result of Bernstein et al. [1] shows that in case of history-dependent policies, the complexity of POSGs is greater than the complexity of POMDPs. We show that this difference does not appear when stationary policies are considered instead of history-dependent policies. For POMDPs, the problem appears to be NP-complete [6]. A stationary policy is a mapping $O \to A$ from observations to actions. Whenever the same observation is made, the same action is chosen by a stationary policy.

**Theorem 3.1** *For any $k \geq 2$, the cooperative agents problem for $k$ agents for stationary policies is* NP-*complete.*

**Proof** We start with proving NP-hardness. A POSG with only one agent is a POMDP. The problem of deciding, for a given POMDP $\mathscr{M}$, whether there exists a stationary policy such that the short-term performance of $\mathscr{M}$ is greater than 0, is NP-complete [6]. Hence, the cooperative agents problem for stationary policies is NP-hard.

It remains to show containment in NP. Let $\mathcal{M} = (I,S,s_0,A,O,t,o,r)$ be a POSG. We assume that $t$ is represented in a straightforward way as a table. Let $\pi_1,\ldots,\pi_k$ be a sequence of stationary policies for the $k$ agents. This sequence can be straightforwardly represented using not more space than the representation of $t$ takes. Under a fixed sequence of policies, the performance of the POSG for all of the agents can be calculated in polynomial time. Using a guess and check approach (guess the stationary policies and evaluate the POSG), this shows that the cooperative agents problem for stationary policies is in NP. $\qquad\square$

In the same way we can characterize the complexity of a problem that we will need in the proof of Lemma 3.3.

**Corollary 3.2** *The following problem is* coNP-*complete.*

> *instance:*    *a POSG $\mathcal{M}$ for k agents*
> *query:*      *do all agents under every stationary policy have positive performance? (I.e.*
>             *$\forall$ stationary $\pi_1\ldots\pi_k : \bigwedge_{i=1}^k perf_i(\mathcal{M},\pi_1^k) > 0$ ?)*

The cooperative agents problem was shown to be NEXP-complete by Bernstein et al. [1]. Not surprisingly, if the agents compete, the problem becomes harder.

**Lemma 3.3** *For every $k \geq 1$, the competing agents problem for $2k$ agents is in* $\text{NEXP}^{\text{NP}}$.

**Proof** The basic idea is as follows. We guess policies $\pi_1,\pi_2,\ldots,\pi_k$ for agents $1,2,\ldots,k$, and construct a POSG that "implements" these policies and leaves open the actions chosen by agents $k+1,\ldots,2k$.

This new POSG has states for all short-term trajectories through the origin POSG. Therefore, its size is exponential in the size of the origin POSG. Because the history is stored in every state, and the POSG is loop-free, it turns out that the new POSG can be taken as a POMDP for which a (joint) policy with positive reward is searched. This problem is known to be NP-complete.

Let $\mathcal{M} = (I,S,s_0,A,O,t,o,r)$ be a POSG with $2k$ agents, and let $\pi_1,\ldots,\pi_k$ be short-term policies for $\mathcal{M}$. We define a $k$-agent POSG $\mathcal{M}' = (I',S',s_0',A,O',t',o',r')$ as follows[2]. In $\mathcal{M}'$, we have as agents those of $\mathcal{M}$, whose policies are not fixed, i.e. $I' = \{k+1,\ldots,2k\}$. The set of states of $\mathcal{M}'$ is the cross product of states from $\mathcal{M}$ and all trajectories up to length $|\mathcal{M}|$ over $S$, i.e. $S' = S \times S^{\leq|\mathcal{M}|+1}$. The meaning of state $(s,u) \in S'$ is, that state $s$ can be reached on a trajectory $u$ (that ends with $s$) through $\mathcal{M}$ with the fixed policies. The initial state $s_0'$ is $s_0' = (s_0,s_0)$. The state $(s_0,\varepsilon)$ is taken as a special sink state. After $|\mathcal{M}|+2$ steps, the sink state is entered in $\mathcal{M}'$ and it is not left thereafter. All rewards gained in the sink state are 0. Now for the transition probabilities. If $s$ is reached on trajectory $u$ in $\mathcal{M}$ and the actions $a_1,\ldots,a_k$ are according to the fixed policies $\pi_1,\ldots,\pi_k$, then the probabiliy of reaching state $s'$ on trajectory $us'$ according to $t$ in $\mathcal{M}$ is the same as to reach $(s',us')$ in $\mathcal{M}'$ from $(s,u)$. In the formal description, the sink state has to be considered, too.

$$t'((s,u),a_k,\ldots,a_{2k},(\hat{s},\hat{u})) =$$
$$\begin{cases} 0, & \text{if } u \neq \varepsilon \text{ and } u\hat{s} \neq \hat{u} \\ t(s,\pi_1(o(us,1)),\cdots,\pi_k(o(us,k)),a_{k+1},\ldots,a_{2k},\hat{s}), & \text{if } \hat{u} = u\hat{s}, |\hat{u}| \leq |\mathcal{M}|, u \neq \varepsilon \\ 1, & \text{if } |u| = |\mathcal{M}|+1 \text{ or } u = \varepsilon, \text{ and } \hat{u} = \varepsilon \end{cases}$$

The observation in $\mathcal{M}'$ is the sequence of observations made in the trajectory that is contained in each state, i.e. $o'((s,w)) = o(w)$, where $o(\varepsilon)$ is any element of $O$. Finally, the rewards. Essentially, we are interested in the rewards obtained by the agents $1,2,\ldots,k$. The rewards obtained by the other agents have no impact on this, only the actions the other agents choose. Therefore, agent $i$ obtains the rewards in $\mathcal{M}'$ that are obtained by agent $i-k$ in $\mathcal{M}$. In this way, the agents $k+1,\ldots,2k$ obtain in $\mathcal{M}'$ the same rewards that are obtained by agents $1,2,\ldots,k$ in $\mathcal{M}$, and this is what we are interested in. This results in $r'((s,u),a_k,\ldots,a_{2k},i) = r(s,\pi_1(o(u,1)),\cdots,\pi_k(o(u,k)),a_{k+1},\ldots,a_{2k},i-k)$ for $i = k+1,\ldots,2k$.

Notice that the size of $\mathcal{M}'$ is exponential in the size of $\mathcal{M}$. The sink state in $\mathcal{M}'$ is the only state that lies on a loop. This means, that on all trajectories through $\mathcal{M}'$, the sink state is the only state that may appear more than once. All states other than the sink state contain the full history of how they are reached. Therefore, there is a one-to-one correspondence between history-dependent policies for $\mathcal{M}$ and stationary policies for $\mathcal{M}'$ (with regard to horizon $|\mathcal{M}|$). Moreover, the corresponding policies have the same performances.

**Claim 1** *Let $\pi_1, \ldots, \pi_{2k}$ be short-term policies for $\mathcal{M}$, and let $\hat{\pi}_{k+1}, \ldots, \hat{\pi}_{2k}$ be their corresponding stationary policies for $\mathcal{M}'$.*

*For $|\mathcal{M}|$ steps and $i = 1, 2, \ldots, k$, $perf_i(\mathcal{M}, \pi_1^{2k}) = perf_{i+k}(\mathcal{M}', \hat{\pi}_{k+1}^{2k})$.*

Thus, this yields an $\mathrm{NEXP}^{\mathrm{NP}}$ algorithm to decide the competitive agents problem. The input is a POSG $\mathcal{M}$ for $2k$ agents. In the first step, the policies for the agents $1, 2, \ldots, k$ are guessed. This takes nondeterministic exponential time. In the second step, the POSG $\mathcal{M}'$ is constructed from the input $\mathcal{M}$ and the guessed policies. This takes exponential time (in the length of the input $\mathcal{M}$). Finally, the oracle is queried whether $\mathcal{M}'$ has positive performance for all agents under all stationary policies. This problem belongs to coNP (Corollary 3.2). Henceforth, the algorithm shows the competing agents problem to be in $\mathrm{NEXP}^{\mathrm{NP}}$. $\qquad\square$

**Lemma 3.4** *For every $k \geq 2$, the competing agents problem for $2k$ agents is hard for $\mathrm{NEXP}^{\mathrm{NP}}$.*

**Proof** We give a reduction from the exponential $\Sigma_2$ square tiling problem to the competing agents problem.

Let $\mathcal{T} = (T, 1^k)$ be an instance of the exponential $\Sigma_2$ square tiling problem, where $T = (V, H)$ is a tile type. We will show how to construct a POSG $\mathcal{M}$ with 4 agents from it, such that $\mathcal{T}$ is a positive instance of the exponential $\Sigma_2$ square tiling problem if and only if (1) agents 1 and 2 have a tiling for the $2^k$ square with final row $w$ such that (2) agents 3 and 4 have no tiling for the $2^k$ square with initial row $w$.

The basic idea for checking of tilings with POSGs for two agents stems from Bernstein et al. [1], but we give a slight simplification of their proof technique, and in fact have to extend it for four agents later on. The POSG is constructed so that on every trajectory each agent sees a position in the square. This position is chosen by the process. The only action of the agent that has impact on the process is putting a tile on the given position. In fact, the same position is observed by the agents in different states of the POSG. From a global point of view, the process splits into two parts. The first part checks whether both agents know the same tiling, without checking that it is a correct tiling. In the state where the agents are asked to put their tiles on the given position, a high negative reward is obtained if the agents put different tiles on that position. "High negative" means that, if there is at least one trajectory on which such a reward is obtained, then the performance of the whole process will be negative. The second part checks whether the tiling is correct. The idea is to give both the agents neighboured positions in the square and to ask each which tile she puts on that position. Notice that the agents do not know in which part of the process they are. This means, that they do not know whether the other agent is asked for the same position, or for its upper or right neighbour. This is why the agents cannot cheat the process. A high negative reward will be obtained if the agents' tiles do not fit together.

For the first part, we need to construct is a POSG $\mathcal{P}_k$ for two agents, that allows both agents to make the same sequence of observations consisting of $2k$ bits. This sequence is randomly chosen, and encodes a position in a $2^k \times 2^k$ grid. At the end, state same is reached, at which no observation is made. At this state, it will be checked whether both agents put the same tile at this position (see later on). The task of $\mathcal{P}_k$ is to provide both agents with the same position. Figure 1 shows an example for a $2^4 \times 2^4$-square. The initial state is $s_4$. Dashed arrows indicate transitions with probability $\frac{1}{2}$ independent of the actions. The observation of agent 1 is written on the left hand side of the states, and the observations of agent 2 at the right hand side. In $s_4$, the agents make no observation. In $\mathcal{P}_k$ both agents always make the same observations.

The second part is more involved. The goal is to provide both agents with neighboured positions in the square. Eventually, it is checked whether the tiles they put on the neighboured positions are according to the tile type $T$. Because the positions are encoded in binary, we can make use

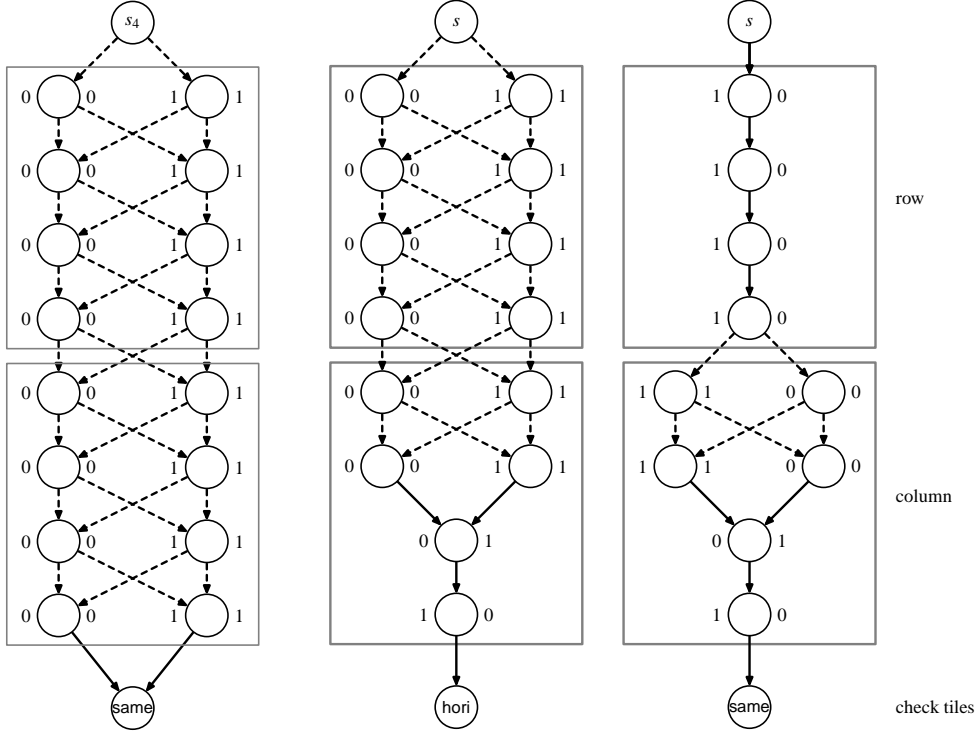

row

column

check tiles

Figure 1: $\mathscr{P}_4$    Figure 2: $\mathscr{C}_{3,4}$    Figure 3: $\mathscr{L}_{3,4}$

of the following fact of subsequent binary numbers. Let $u = u_1 \ldots u_k$ and $w = w_1 \ldots w_k$ be bitwise representation of strings. if $n_w = n_u + 1$, then for some index $l$ it holds that (1) $u_i = w_i$ for $i = 1, 2, \ldots, l-1$, (2) $w_l = 1$ and $u_l = 0$, and (3) $w_j = 0$ and $u_j = 1$ for $j = l+1, \ldots, k$.

The POSG $\mathscr{C}_{l,k}$ is intended to provide the agents with two neighboured positions in the same row, where the index of the leftmost bit of the column encoding where both positions distinguish is $l$. (The $\mathscr{C}$ stands for *column*.) Figure 2 shows an example for the $2^4$-square. The "final state" of $\mathscr{C}_{l,k}$ is the state hori, from which it is checked whether the agents put horizontally fitting tiles together.

In the same way, a POSG $\mathscr{R}_{l,k}$ can be constructed ($\mathscr{R}$ stands for *row*), whose task is, to check whether two tiles in neighboured rows correspond to a correct tiling. This POSG has the final state vert, from which on it is checked whether two tiles fit vertically.

Finally, we have to construct the last part of the POSG. It consists of the states same, hori, vert (as mentioned above), good, bad, and sink. All transitions between these states are deterministic (i.e. with probability 1). From state same the state good is reached, if both agents take the same action – otherwise bad is reached. From state hori the state good is reached, if action $a_1$ by agent 1 and $a_2$ by agent 2 make a pair $(a_1, a_2)$ in $H$, i.e. in the set of horizontally correct pairs of tiles – otherwise bad is reached. Similarly, from state vert the state good is reached, if action $a_1$ by agent 1 and $a_2$ by agent 2 make a pair $(a_1, a_2)$ in $V$. All these transitions are with reward 0. From state good the state sink is reached on every action with reward 1, and from state bad the state sink is reached on every action with reward $-(2^{2k+2})$. When the state sink is reached, the process stays there on any action, and all agents obtain reward 0. All rewards are the same for both agents. (This part can be seen in the overall picture in Figure 4).

From these POSGs we construct a POSG $\mathscr{T}_{2,k}$ that checks whether two agents know the same correct tiling for a $2^k \times 2^k$ square, as described above. There are $2k+1$ parts of $\mathscr{T}_{2,k}$. The initial state of each part can be reached with one step from the initial state $s_0$ of $\mathscr{T}_{2,k}$. The parts of $\mathscr{T}_{2,k}$ are as follows.

- $\mathscr{P}_{2k}$ with initial state $s$ (checks whether two agents have the same tiling)
- For each $l = 1, 2, \ldots, k$, we take $\mathscr{C}_{l,k}$. Let $c_l$ be the initial state of $\mathscr{C}_{l,k}$.

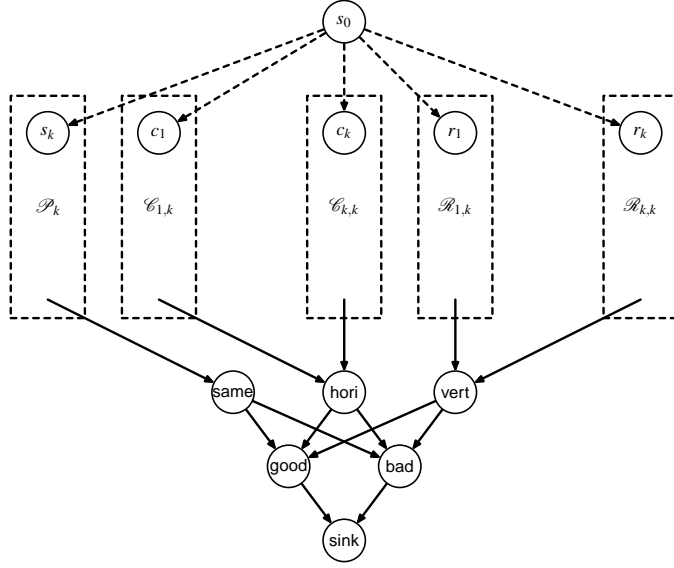

Figure 4: $\mathscr{T}_{2,k}$

- For each $l = 1, 2, \ldots, k$, we take $\mathscr{R}_{l,k}$. Let $r_l$ be the initial state of $\mathscr{R}_{l,k}$.

There are $2^{2k} + 2 \cdot \sum_{l=1}^{k} 2^k \cdot 2^{l-1} =: tr(k)$ trajectories with probability $> 0$ through $\mathscr{T}_{2,k}$. Notice that $tr(k) < 2^{2k+2}$. From the initial state $s_0$ of $\mathscr{T}_{2,k}$, each of the initial states of the parts is reachable independent on the action chosen by the agents. We will give transition probabilities to the transition from $s_0$ to each of the initial states of the parts in a way, that eventually each trajectory has the same probability.

$$t(s_0, a_1, a_2, s') = \begin{cases} \frac{2^{2k}}{tr(k)}, & \text{if } s' = s, \text{ i.e. the initial state of } \mathscr{P}_k \\ \frac{2^{k+l-1}}{tr(k)} & \text{if } s \in \{r_l, c_l \mid l = 1, 2, \ldots, k\} \end{cases}$$

In the initial state $s_0$ and in the initial states of all parts, the observation $\varepsilon$ is made. When a state same, hori, vert is reached, each agent has made $2k + 3$ observations, where the first and last are $\varepsilon$ and the remaining $2k$ are each in $\{0, 1\}$. Such a state is the only one where the actions of the agents have impact on the process. Because of the partial observability, they cannot know in which part of $\mathscr{T}_{2,k}$ they are. The agents can win, if they both know the same correct tiling and interpret the sequence of observations as the position in the grid they are asked to put a tile on. On the other hand, if both agents know different tilings or the tiling they share is not correct, then at least one trajectory will end in a bad state and has reward $-(2^{2k+2})$. The structure of the POSG is given in Figure 4.

**Claim 2** *Let $(T, 1^k)$ be an instance of the exponential square tiling problem.*

*(1) There exists a polynomial time algorithm, that on input $(T, 1^k)$ outputs $\mathscr{T}_{2,k}$.*

*(2) There exists a $T$-tiling of the $2^k$ square if and only if there exist policies for the agents under which $\mathscr{T}_{2,k}$ has performance $> 0$.*

Part (1) is straightforward. Part (2) is not much harder. If there exists a $T$-tiling of the $2^k$ square, both agents use the same policy according to this tiling. Under these policies, state bad will not be reached. This guarantess performance $> 0$ for both agents. For the other direction: if there exist policies for the agents under which $\mathscr{T}_{2,k}$ has performance $> 0$, then state bad is not reached. Hence, both agents use the same policy. It can be shown inductively that this policy "is" a $T$-tiling of the $2^k$ square.

The POSG for the competing agents problem with 4 agents consists of three parts. The first part is a copy of $\mathscr{T}_{2,k}$. It is used to check whether the first square can be tiled correctly (by agents 1 and 2). In this part, the negative rewards are increased in a way that guarantees the performance of the POSG to be negative whenever agents 1 and 2 do not correctly tile their square. The second part is a modified copy of $\mathscr{T}_{2,k}$. It is used to check whether the second square can be tiled correctly (by agents 3 and 4). Whenever state bad is left in this copy, reward 0 is obtained, and whenever state good is left, reward $-1$ is obtained. The third part checks whether agent 1 puts the same tiles into the last row of its square as agent 3 puts into the first row of its square. (See $\mathscr{L}_{3,4}$ in Figure 3 as an example.) If this succeeds, the performance of the third part equals 0, otherwise it has performance 1. These three parts run in parallel.

If agents 1 and 2 have a tiling for the first square, the performance of the first part equals 1.

- If agents 3 and 4 are able to continue this tiling through their square, the performance of the second part equals $-1$ and the performance of the third part equals 0. At all, the performance of the POSG under these policies equals 0.

- If agents 3 and 4 are not able to continue this tiling through their square, then the performance of part 2 and part 3 is strictly greater $-1$. At all, the performance of the POSG under these policies is $> 0$.

$\square$

Lemmas 3.3 and 3.4 together yield completeness of the competing agents problem.

**Theorem 3.5** *For every $k \geq 2$, the competing agents problem for $2k$ agents is complete for* $\mathrm{NEXP}^{\mathrm{NP}}$.

## 4   Conclusion

We have shown that competition makes life—and computation—more complex. However, in order to do so, we needed teamwork. It is not yet clear what the complexity is of determining the existence of a good strategy for Player I in a 2-person POSG, or a 1-against-many POSG.

There are other variations that can be shown to be complete for $\mathrm{NEXP}^{\mathrm{NP}}$, a complexity class that, shockingly, has not been well explored. We look forward to further results about the complexity of POSGs, and to additional $\mathrm{NEXP}^{\mathrm{NP}}$-completeness results for familiar AI and ML problems.

## Footnotes

[1]The size of the representation of $\mathcal{M}$ is the number of bits to encode the entire model, where the function $t$, $o$, and $r$ are encoded by tables. We do *not* consider smaller representations. In fact, smaller representations may increase the complexity.

[2]$S^{\leq|\mathcal{M}|}$ denotes the set of sequences up to $|\mathcal{M}|$ elements from $S$. The empty sequence is denoted by $\varepsilon$. For $w \in S^{\leq|\mathcal{M}|}$ we use $o(w,i)$ to describe the sequence of observations made by agent $i$ on trajectory $w$. The concatenation of sequences $u$ and $w$ is denoted $uw$. We do not distinguish between elements of sets and sequences of one element.

## References

[1] Daniel S. Bernstein, Robert Givan, Neil Immerman, and Shlomo Zilberstein. The complexity of decentralized control of Markov decision processes. *Math. Oper. Res.*, 27(4):819–840, 2002.

[2] E. Hansen, D. Bernstein, and S. Zilberstein. Dynamic programming for partially observable stochastic games. In *Proceedings of the Nineteenth National Conference on Artificial Intelligence (AAAI-04)*, pages 709–715, 2004.

[3] Hao Wang. Proving theorems by pattern recognition II. *Bell Systems Technical Journal*, 40:1–42, 1961.

[4] M. Savelsbergh and P. van Emde Boas. Bounded tiling, an alternative to satisfiability. In Gerd Wechsung, editor, *2nd Frege Conference*, volume 20 of *Mathematische Forschung*, pages 354–363. Akademie Verlag, Berlin, 1984.

[5] C.H. Papadimitriou and J.N. Tsitsiklis. The complexity of Markov decision processes. *Mathematics of Operations Research*, 12(3):441–450, 1987.

[6] Martin Mundhenk, Judy Goldsmith, Christopher Lusena, and Eric Allender. Complexity results for finite-horizon Markov decision process problems. *Journal of the ACM*, 47(4):681–720, 2000.

